# Psychiatry: insights into depression through normative decision-making models

**Quentin JM Huys**[1,2,*]  **Joshua T Vogelstein**[3,*]  **and Peter Dayan**[2,*]

[1]Center for Theoretical Neuroscience, Columbia University, New York, NY 10032, USA
[2]Gatsby Computational Neuroscience Unit, University College London, London, WC1N 3AR, UK
[3]Johns Hopkins School of Medicine, Baltimore MD 21231, USA

## Abstract

Decision making lies at the very heart of many psychiatric diseases. It is also a central theoretical concern in a wide variety of fields and has undergone detailed, in-depth, analyses. We take as an example Major Depressive Disorder (MDD), applying insights from a Bayesian reinforcement learning framework. We focus on anhedonia and helplessness. Helplessness—a core element in the conceptualizations of MDD that has lead to major advances in its treatment, pharmacological and neurobiological understanding—is formalized as a simple prior over the outcome entropy of actions in uncertain environments. Anhedonia, which is an equally fundamental aspect of the disease, is related to the effective reward size. These formulations allow for the design of specific tasks to measure anhedonia and helplessness behaviorally. We show that these behavioral measures capture explicit, questionnaire-based cognitions. We also provide evidence that these tasks may allow classification of subjects into healthy and MDD groups based purely on a behavioural measure and avoiding any verbal reports.

There are strong ties between decision making and psychiatry, with maladaptive decisions and behaviors being very prominent in people with psychiatric disorders. Depression is classically seen as following life events such as divorces and job losses. Longitudinal studies, however, have revealed that a significant fraction of the stressors associated with depression do in fact follow MDD onset, and that they are likely due to maladaptive behaviors prominent in MDD (Kendler et al., 1999). Clinically effective 'talking' therapies for MDD such as cognitive and dialectical behavior therapies (DeRubeis et al., 1999; Bortolotti et al., 2008; Gotlib and Hammen, 2002; Power, 2005) explicitly concentrate on altering patients' maladaptive behaviors and decision making processes.

Decision making is a promising avenue into psychiatry for at least two more reasons. First, it offers powerful analytical tools. Control problems related to decision making are prevalent in a huge diversity of fields, ranging from ecology to economics, computer science and engineering. These fields have produced well-founded and thoroughly characterized frameworks within which many issues in decision making can be framed. Here, we will focus on framing issues identified in psychiatric settings within a normative decision making framework.

Its second major strength comes from its relationship to neurobiology, and particularly those neuro-modulatory systems which are powerfully affected by all major clinically effective pharmacotherapies in psychiatry. The understanding of these systems has benefited significantly from theoretical accounts of optimal control such as reinforcement learning (Montague et al., 1996; Kapur and Remington, 1996; Smith et al., 1999; Yu and Dayan, 2005; Dayan and Yu, 2006). Such accounts may be useful to identify in more specific terms the roles of the neuromodulators in psychiatry (Smith et al., 2004; Williams and Dayan, 2005; Moutoussis et al., 2008; Dayan and Huys, 2008).

---

[*]qhuys@cantab.net, joshuav@jhu.edu, dayan@gatsby.ucl.ac.uk; www.gatsby.ucl.ac.uk/∼qhuys/pub.html

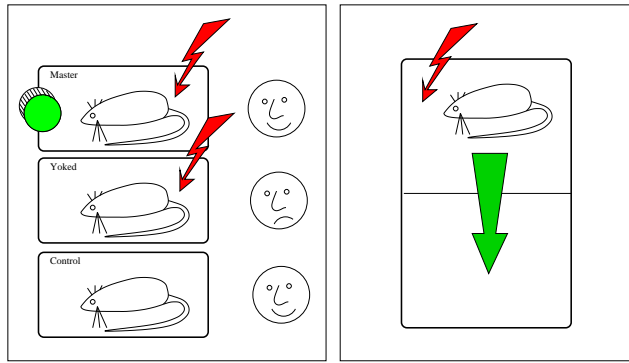

Figure 1: The learned helplessness (LH) paradigm. Three sets of rats are used in a sequence of two tasks. In the first task, rats are exposed to escapable or inescapable shocks. Shocks come on at random times. The master rat is given escapable shocks: it can switch off the shock by performing an action, usually turning a wheel mounted in front of it. The yoked rat is exposed to precisely the same shocks as the master rat, i.e its shocks are terminated when the master rat terminates the shock. Thus its shocks are inescapable, there is nothing it can do itself to terminate them. A third set of rats is not exposed to shocks. Then, all three sets of rats are exposed to a shuttlebox escape task. Shocks again come on at random times, and rats have to shuttle to the other side of the box to terminate the shock. Only yoked rats fail to acquire the escape response. Yoked rats generally fail to acquire a wide variety of instrumental behaviours, either determined by reward or, as here, by punishment contingencies.

This paper represents an initial attempt at validating this approach experimentally. We will frame core notions of MDD in a reinforcement learning framework and use it to design behavioral decision making experiments. More specifically, we will concentrate on two concepts central to current thinking about MDD: anhedonia and learned helplessness (LH, Maier and Seligman 1976; Maier and Watkins 2005). We formulate helplessness parametrically as prior beliefs on aspects of decision trees, and anhedonia as the effective reward size. This allows us to use choice behavior to infer the degree to which subjects' behavioral choices are characterized by either of these. For validation, we correlate the parameters inferred from subjects' behavior with standard, questionnaire-based measures of hopelessness and anhedonia, and finally use the inferred parameters alone to attempt to recover the diagnostic classification.

## 1 Core concepts: helplessness and anhedonia

The basic LH paradigm is explained in figure 1. Its importance is manifold: the effect of inescapable shock on subsequent learning is sensitive to most classes of clinically effective antidepressants; it has arguably been a motivation framework for the development of the main talking therapies for depression (cognitive behavioural therapy, Williams (1992), it has motivated the development of further, yet more specific animal models (Willner, 1997), and it has been the basis of very specific research into the cognitive basis of depression (Peterson et al., 1993).

Behavioral control is the central concept in LH: yoked and master rat do not differ in terms of the amount of shock (stress) they have experienced, only in terms of the behavioural control over it. It is not a standard notion in reinforcement learning, and there are several ways one could translate the concept into RL terms. At a simple level, there is intuitively more behavioural control if, when repeating one action, the same outcome occurs again and again, than if this were not true. Thus, at a very first level, control might be related to the outcome entropy of actions (see Maier and Seligman 1976 for an early formulation). Of course, this is too simple. If all available actions deterministically led to the same outcome, the agent has very little control. Finally, if one were able to achieve all outcomes except for the one one cares about (in the rats' case switching off or avoiding the shock), we would again not say that there is much control (see Huys (2007); Huys and Dayan (2007) for a more detailed discussion). Despite its obvious limitations, we will here concentrate on the simplest notion for reasons of mathematical expediency.

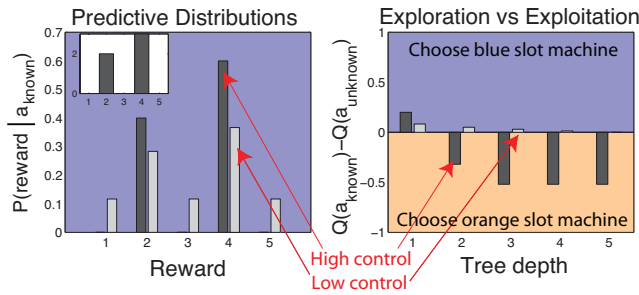

Figure 2: Effect of $\gamma$ on predictions, $\mathcal{Q}$-values and exploration behaviour. Assume a slot machine (blue) has been chosen five times, with possible rewards 1-5, and that reward 2 has been obtained twice, and reward 4 three times (inset in left panel). **Left**: Predictive distribution for a prior with negative $\gamma$ (low control) in light gray, and large $\gamma$ (extensive control) in dark gray. We see that, if the agent believes he has much control (and outcome distributions have low entropy), the predictive distribution puts all mass on the observations. **Right**: Assume now the agent gets up to 5 more pulls (tree depth 1-5) between the blue slot machine and a new, orange slot machine. The orange slot machine's predictive distribution is flat as it has never been tried, and its expected value is therefore 3. The plot shows the difference between the values for the two slot machines. First consider the agent only has one more pull to take. In this case, independently of the priors about control, the agent will choose the blue machine, because it is just slightly better than average. Note though that the difference is more pronounced if the agent has a high control prior. But things change if the agent has two or more choices. Now, it is worth trying out the new machine *if* the agent has a high-control prior. For in that case, if the new machine turns out to yield a large reward on the first try, it is likely to do so again for the second and subsequent times. Thus, the prior about control determines the exploration bonus.

The second central concept in current conceptions of MDD is that of reward sensitivity. Anhedonia, an inability to enjoy previously enjoyable things, is one of two symptoms necessary for the diagnosis of depression (American Psychiatric Association, 1994). A number of tasks in the literature have attempted to measure reward sensitivity behaviourally. While these generally concur in finding decreased reward sensitivity in subjects with MDD, these results need further clarification. Some studies show interactions between reward and punishment sensitivities with respect to MDD, but important aspects of the tasks are not clearly understood. For instance, Henriques et al. (1994); Henriques and Davidson (2000) show decreased resonsiveness of MDD subjects to rewards, but equally show decreased resonsiveness of healthy subjects to punishments. Pizzagalli et al. (2005) introduced an asymmetrically rewarded perceptual discrimination task and show that the rate of change of the response bias is anticorrelated with subjects' anhedonic symptoms. Exactly how decreased reward responsivity can account for this is at pressent not clear.

Great care has to be taken to disentangle these two concepts. Anhedonia and helplessness both provide good reasons for not taking an action: either because the reinforcements associated with the action are insufficient (anhedonia), or because the outcome is not judged a likely result of taking some particular action (if actions are thought to have large outcome entropy).

## 2 A Bayesian formulation of control

We consider a scenario where subjects have no knowledge of the outcome distributions of actions, but rather learn about them. This means that their prior beliefs about the outcome distributions are not overwhelmed by the likelihood of observations, and may thus have measurable effects on their action choices. In terms of RL, this means that agents do not know the decision tree of the problem they face. Control is formulated as a prior distribution on the outcome distributions, and thereby as a prior distribution on the decision trees.

The concentration parameter $\alpha$ of a Dirichlet process can very simply parametrise entropy, and, if used as a prior, allow for very efficient updates of the predictive distributions of actions. Let us assume we have actions $\mathcal{A}$ which have as outcomes rewards $\mathcal{R}$, and keep count $N_t(r, a) =$

$\sum_{k:k<t;a_k=a} \delta_{r,r_k}$ of the number of times a particular reward $r \in \mathcal{R}$ was observed for each action $a \in \mathcal{A}$, where $t$ is the number of times that action has been chosen, $r_t$ is the reward on the $t^{\text{th}}$ trial and $\delta$ is the Kronecker delta. The predictive distribution for action $a$ is then

$$P(r|N_t, a, \alpha) = \frac{\alpha}{\alpha + N_t(a)} B(r) + \frac{1}{\alpha + N_t(a)} N_t(r, a) \tag{1}$$

Here, $B(r)$ is the base distribution, which we assume is flat, and $N_t(a) = \sum_r N_t(r, a)$ is the number of times action $a$ was chosen up to trial $t$. Thus, the first time an action is chosen, we draw a sample from $B(r)$. For $\alpha = 0$, we then always draw that very same sample again. For $\alpha = \infty$, we keep drawing from the same flat outcome distribution. Thus, $\alpha$ very simply determines the entropy of the actions' outcome distribution. To match parametric values onto control more intuitively, let $\gamma = -\log(\alpha)$ be the control parameter.

The action choice problem is now to choose action $a = \text{argmax}_a \mathcal{Q}(a|N)$, where the $\mathcal{Q}$ values are defined by the Bellman equation of our problem:

$$\mathcal{Q}_t(a|N_t, \gamma) = \sum_r p(r|N_t, a, \gamma)[r + \text{argmax}_{a'} \mathcal{Q}_t(a'|N_{t+1}(r), \gamma)] \tag{2}$$

where $N_{t+1}(r)$ is the count including the (anticipated) reward $r$. The effect of the parameter $\gamma$ on $\mathcal{Q}$ values is illustrated in Figure 2. One can now infer the maximum likelihood (ML) parameters of the prior by writing the probability of the subject's observed actions as a standard softmaxed version of the $\mathcal{Q}$ values:

$$\{\hat{\gamma}, \hat{\beta}\}_{ML} = \text{argmax}_{\gamma, \beta} \prod_t p(a_t|N_t, \gamma, \beta) \tag{3}$$

$$\text{where} \quad p(a_t|N_t, \gamma, \beta) = \frac{\exp(\beta \mathcal{Q}_t(a|N_t, \gamma))}{\sum_{a'} \exp(\beta \mathcal{Q}_t(a'|N_t, \gamma))} \tag{4}$$

where we have introduced a second parameter $\beta$, which is either the softmax inverse temperature, or, alternatively and equivalently, the size of the rewards (the maximum of $R$ and $\beta$ are not both inferable from action observations only).

Simulations of the inference revealed that the parameters $\gamma$ and $\beta$, our inferred reward sensitivity and prior on control, were correlated. To alleviate this problem, subjects were additionally given a reward sensitivity task which was interleaved with the control task (see below for the task descriptions). The structure of the reward sensitivity task is such that $Q$ values are correctly defined by a Rescorla-Wagner (RW) learning rule:

$$\mathcal{Q}_t^{RW}(a) = (1 - \epsilon)\mathcal{Q}_{t-1}^{RW}(a) + \epsilon r_t \tag{5}$$

where $\mathcal{Q}_t^{RW}(a)$ is the $\mathcal{Q}$ value of action $a$ at choice $t$, $\epsilon$ is the learning rate, and actions probabilities again defined via softmax with a parameter $\beta$ as in equation 4. Note, importantly, i) that this is not dependent on $\gamma$, the prior belief about control and ii) that unlike equation 2 above, this is a 'model-free' algorithm that does not look ahead and thus does not take anticipated rewards into account). Combining inference in the two tasks (sharing $\beta$ between them), allows us to use the reward sensitivity task as a prior on $\beta$ for the control task and to eliminate the correlation.

## 2.1 Task and subjects

**Control task:** The effects illustrated in figure 2 are easily elicited in a simple behavioral task. Subjects are told to imagine that they are in a large casino, and will be dropped randomly in each of 100 rooms. In each room, they will get to choose between slot machines. At first, they see only one slot machine, which they have to choose. Next, they get to choose between two slot machines. A new machine is presented whenever all machines on the screen have been tried. Thus, the exploratory drive is always maintained with one unexplored slot machine. Subjects get 8 choices per room, and thus get to try a maximum of 8 machines once in each room. Subjects are informed that outcomes for each slot machine are between 0 and 9 points. Overall, subjects are thus always kept in the dark about the true outcome distribution of any one slot machine. Thus, their prior beliefs become relevant. For healthy control subjects, one room was chosen randomly and the total number of points

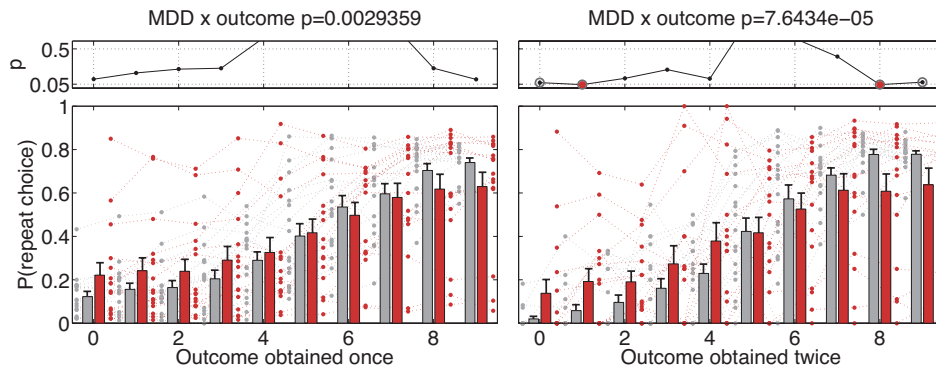

Figure 3: Repeat modulation. Bottom plots: Probability of choosing a slot machine again given that it has just yielded a particular outcome. Control subjects are in gray and MDD subjects in red (all individuals as dots, means ± 1 std. err. as bars; red dots on the right of bars, gray dots on the left of bars). Top plots: uncorrected p-values comparing the two groups for every individual outcome. Left panel: after observing a particular outcome once, Right panel: after observing the same outcome on a particular machine twice in a row. The p-values at the top indicate the ANOVA interaction of outcome size with group. Thus, we see here that subjects with MDD are more likely to stick with a bad machine, and more likely to move away from a good machine. The same result is observed when fitting sigmoids to each subject and comparing the inferred parameters (data not shown).

earned in that room determined the payment (1 point = 1 US$, minimum 10US$, maximum 50US$). MDD subjects were given the same instructions, but, for ethical reasons, could not be paid.

**Reward sensitivity task:** Subjects chose repeatedly (300 times) between two stacks of cards with probabilistic binary outcomes. The underlying probabilities of a reward changed as a (squashed) Ornstein-Uhlenbeck process. This task was thus accurately described by a standard Rescorla-Wagner (RW) rule (Daw et al., 2006).

**Questionnaire measures:** Finally, each subject filled out two questionnaires: the Beck Helplessness Score (BHS), and the Beck Depression Inventory (BDI) which are standard questionnaire measures of hopelessness and anhedonia respectively. We extracted the anhedonic subcomponent, BDIa, as the sum of responses on questions 4, 12, 15 and 21 of the BDI.

**Subjects:** We recruited 17 healthy control subjects from the community. 15 subjects with MDD were recruited as part of an ongoing treatment study, and asked to take the behavioural test while waiting to see the psychiatrists. All subjects were given a full Structured Clinical Interview for DSM-IV (First et al., 2002a,b). All MDD subjects met criteria for a current major depressive episode. Three subjects had additionally a diagnosis of either Panic Disorder (2) or Bipolar Disorder II (1). All the healthy control subjects had neither a present psychiatric disorder, nor a history thereof. All procedures were approved by the New York State Institue of Psychiatry Institutional Review Board. The subjects were matched for sex and educational level, but not for age. We thus included age in our model formulations to exclude its effects as a nuisance variable. The depressed sample was older, but throughout, the effects of age correlate *negatively* with those of depression.

## 3 Results

### 3.1 Reward sensitivity

**Preliminary analysis:** Repeat modulation, a very simple proxy measure of choices, provides a first glimpse at the effects of depression on the first task. Figure 3 shows the probability with which subjects chose a slot machine again after having received outcomes 0-9. As groups, MDD subjects both avoid bad and exploit good machines less. Nearly half the subjects with MDD show very little modulation with rewards. As a group, MDD subjects appear less sensitive to the reward structure in the first task.

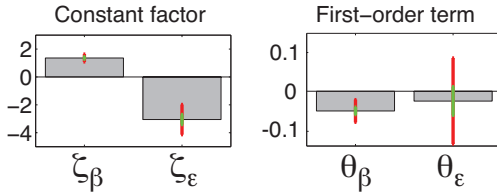

Figure 4: ML inferred values for constant and relevant first-order factors. The green lines are the standard deviations around the ML value, and the red represent three times that. Thus, while BDI is related to the effective reward size $\beta$, it is not related to the learning rate $\epsilon$. Note that here the effect of age has already been accounted for.

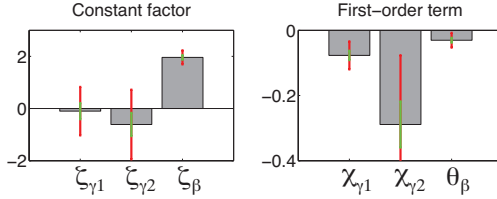

Figure 5: ML inferred values for the constant the relevant first-order factors as in the previous figure. The green lines are the standard deviation around the ML values and the red represent three times that. Thus, the effect of the BHS on control is captured by $\gamma$ that of BDIa on reward sensitivity is captured by $\beta$ as predicted.

**Reward sensitivity:** The main hypothesis with respect to reward sensitivity is that subjects' empirically observed reward sensitivity $\beta$ in equation 5 is inversely related to their expressed anhedonia, BDIa, in the questionnaires. We can build this into the action choice model by parametrising $\beta$ in the $Q^{RW}$ value (equation 5) above explicitly as a function of the questionnaire anhedonia score $BDIa$:

$$\beta(BDIa, AGE) = \theta_\beta BDIa + c_\beta AGE + \zeta_\beta$$

If the hypothesis is true and subjects with higher BDIa scores do indeed care less about rewards, we should observe that $\theta_\beta < 0$. Here, we included a regressor for the $AGE$ as that was a confounding variable in our subject sample. Furthermore, if it is true that anhedonia, as expressed by the questionnaire, relates to reward sensitivity specifically, we should be able to write a similar regression for the learning rate $\epsilon$ (from equation 5)

$$\epsilon(BDIa, AGE) = \theta_\epsilon BDIa + c_\epsilon AGE + \zeta_\epsilon$$

but find that $\theta_\epsilon$ is not different from zero. Figure 4 shows the ML values for the parameters of interest (emphasized in blue in the equations) and confirms that people who express higher levels of anhedonia do indeed show less reward sensitivity, but do not differ in terms of learning rate. If it were the case that subjects with higher BDIa score were just less attentive to the task, one might also expect an effect of BDIa on learning rate.

## 3.2 Control

**Validation:** The control task is new, and we first need to ascertain that subjects were indeed sensitive to main features of the task. We thus fit both a RW-learning rule (as in the previous section, but adjusted for the varying number of available actions), and the full control model. Importantly, both these models have two parameters, but only the full control model has a notion of outcome entropy, and evaluations a tree. The chance probability of subjects' actions was 0.37, meaning that, on average, there were just under three machines on the screen. The probability of the actions under the RW-learning rule was better at 0.48, and that of the full control model 0.54. These differences are highly significant as the total number of choices is 29600. Thus, we conclude that subjects were indeed sensitive to the manipulation of outcome entropy, and that they did look ahead in a tree.

**Prior belief about control:** Applying the procedure from the previous task to the main task, we write the main parameters of equations 2 and 4 as functions of the questionnaire measures and infer linear parameters:

$$\begin{aligned}
\gamma_1(BDIa, BHS, age) &= \chi_{\gamma 1} BHS + \theta_{\gamma 1} BDIa + c_{\gamma 1} AGE + \zeta_{\gamma 1} \\
\gamma_2(BDIa, BHS, age) &= \chi_{\gamma 2} BHS + \theta_{\gamma 2} BDIa + c_{\gamma 2} AGE + \zeta_{\gamma 2} \\
\beta(BDIa, BHS, age) &= \chi_{\beta} BHS + \theta_{\beta} BDIa + c_{\beta} AGE + \zeta_{\beta}
\end{aligned}$$

Importantly, because the BDIa scores and the BHS scores are correlated in our sample (they tend to be large for the subjects with MDD), we include the cross-terms $(\theta_{\gamma 1}, \theta_{\gamma 2}, \chi_\gamma)$, as we are interested in the specific effects of BDIa on $\beta$, as before, and of BHS on $\gamma$.

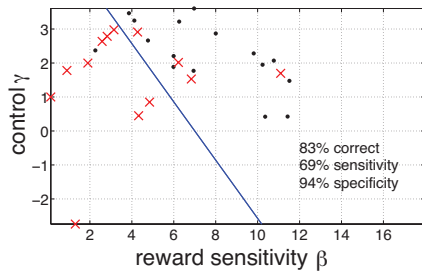

Figure 6: Classification. Controls are shown as black dots, and depressed subjects as red crosses. The blue line is a linear classifier. Thus, the patients and controls can be approximately classified purely on the basis of behaviour.

We here infer and display two separate values $\gamma_1$ and $\gamma_2$. These correspond to the level of control in the first and the second half of the experiment. In fact, to parallel the LH experiments better, the slot machines in the first 50 rooms were actually very noisy (low true $\gamma$), which means that subjects were here exposed to low levels of control just like the yoked rats in the original experiment. In the second half of the experiment on the other hand, slot machines tended to be quite reliable (high true $\gamma$).

Figure 5 shows again the ML values for the parameters of interest (emphasized in blue in the equations). Again, we find that our parameter estimate are very significantly different from zero ($>$ three standard deviations).

The effect of the BHS score on the prior beliefs about control $\gamma$ is much stronger in the second half than of the experiment in the first half, i.e. the effect of BHS on the prior belief about control is particularly prominent when subjects are in a high-control environment and have previously been exposed to a low-control environment. This is an interesting parallel to the learned helplessness experiments in animals.

### 3.3 Classification

Finally we combine the two tasks. We integrate out the learning rate $\epsilon$, which we had found not be related to the questionnaire measures (c.f. figure 4), and use the distribution over $\beta$ from the first task as a prior distribution on $\beta$ for the second task. We also put weak priors on $\gamma$ and infer both $\beta$ and $\gamma$ for the second task on a subject-by-subject basis. Figure 6 shows the posterior values for $\gamma$ and $\beta$ for MDD and healthy subjects and the ability of a linear classifier to classify them.

## 4   Discussion

In this paper, we have attempted to provide a specific formulation of core psychiatric concepts in reinforcement learning terms, i.e. hopelessness as a prior belief about controllability, and anhedonia as reward sensitivity. We have briefly explained how we expect these formulations to have effect in a behavioural situation, have presented a behavioral task explicitly designed to be sensitive to our formulations, and shown that people's verbal expression of hopelessness and anhedonia do have specific behavioral impacts. Subjects who express anhedonia display insensitivity to rewards and those expressing hopelessness behave as if they had prior beliefs that outcome distributions of actions (slot machines) are very broad. Finally, we have shown that these purely behavioural measures are also predictive of their psychiatric status, in that we were able to classify patients and healthy controls purely on the basis of performance.

Several aspects of this work are novel. There have been previous attempts to map aspects of psychiatric dysfunction onto specific parametrizations (Cohen et al., 1996; Smith et al., 2004; Williams and Dayan, 2005; Moutoussis et al., 2008), but we believe that our work represents the first attempt to a) apply it to MDD; b) make formal predictions about subject behavior c) present strong evidence linking anhedonia specifically to reward insensitivity across two tasks d) combine tasks to tease helplessness and anhedonia apart and e) to use the behavioral inferences for classification. The latter point is particularly important, as it will determine any potential clinical significance (Veiel, 1997). In the future, rather than cross-validating with respect to say DSM-IV criteria, it may also be important to validate measures such as ours in their own right in longitudinal studies.

Several important caveats do remain. First, the populations are not fully matched for age. We included age as an additional regressor and found all results to be robust. Secondly, only the healthy subjects were remunerated. However, repeating the analyses presented using only the MDD subjects yields the same results (data not shown). Thirdly, we have not yet fully mirrored the LH experiments. We have so far only tested the transfer from a low-control environment to a high-control environment. To make statements like those in animal *learned* helplessness experiments, the transfer from high-control to low-control environments will need to be examined, too. Fourth, the notion of control we have used is very simple, and more complex notions should certainly be tested (see Dayan and Huys 2008). Fifth, and maybe most importantly, we have so far only attempted to classify MDD and healthy subjects, and can thus not yet make any statements about the specificity of these effects with respect to MDD. Finally, it will be important to replicate these results independently, and possibly in a different modality. Nevertheless, we believe these results to be very encouraging.

**Acknowledgments**: This work would not have been possible without the help of Sarah Hollingsworth Lisanby, Kenneth Miller and Ramin V. Parsey. We would also like to thank Nathaniel Daw and Hanneke EM Den Ouden and René Hen for invaluable discussions. Support for this work was provided by the Gatsby Charitable Foundation (PD), a UCL Bogue Fellowship and the Swartz Foundation (QH) and a Columbia University startup grant to Kenneth Miller.

# References

American Psychiatric Association (1994). *Diagnostic and Statistical Manual of Mental Disorders*. American Psychiatric Association Press.

Bortolotti, B., Menchetti, M., Bellini, F., Montaguti, M. B., and Berardi, D. (2008). Psychological interventions for major depression in primary care: a meta-analytic review of randomized controlled trials. *Gen Hosp Psychiatry*, 30(4):293–302.

Cohen, J. D., Braver, T. S., and O'Reilly, R. C. (1996). A computational approach to prefrontal cortex, cognitive control and schizophrenia: recent developments and current challenges. *Philos Trans R Soc Lond B Biol Sci*, 351(1346):1515–1527.

Daw, N. D., O'Doherty, J. P., Dayan, P., Seymour, B., and Dolan, R. J. (2006). Cortical substrates for exploratory decisions in humans. *Nature*, 441(7095):876–879.

Dayan, P. and Huys, Q. J. M. (2008). Serotonin, inhibition, and negative mood. *PLoS Comput Biol*, 4(2):e4.

Dayan, P. and Yu, A. J. (2006). Phasic norepinephrine: a neural interrupt signal for unexpected events. *Network*, 17(4):335–350.

DeRubeis, R. J., Gelfand, L. A., Tang, T. Z., and Simons, A. D. (1999). Medications versus cognitive behavior therapy for severely depressed outpatients: mega-analysis of four randomized comparisons. *Am J Psychiatry*, 156(7):1007–1013.

First, M. B., Spitzer, R. L., Gibbon, M., and Williams, J. B. (2002a). *Structured Clinical Interview for DSM-IV-TR Axis I Disorders, Research Version, Non-Patient Edition. (SCID-I/NP)*. Biometrics Research, New York State Psychiatric Institute.

First, M. B., Spitzer, R. L., Gibbon, M., and Williams, J. B. (2002b). *Structured Clinical Interview for DSM-IV-TR Axis I Disorders, Research Version, Patient Edition. (SCID-I/P)*. Biometrics Research, New York State Psychiatric Institute.

Gotlib, I. H. and Hammen, C. L., editors (2002). *Handbook of Depression*. The Guilford Press.

Henriques, J. B. and Davidson, R. J. (2000). Decreased responsiveness to reward in depression. *Cognition and Emotion*, 14(5):711–24.

Henriques, J. B., Glowacki, J. M., and Davidson, R. J. (1994). Reward fails to alter response bias in depression. *J Abnorm Psychol*, 103(3):460–6.

Huys, Q. J. M. (2007). *Reinforcers and control. Towards a computational ætiology of depression*. PhD thesis, Gatsby Computational Neuroscience Unit, UCL, University of London.

Huys, Q. J. M. and Dayan, P. (2007). A bayesian formulation of behavioral control. *Under Review*, 0:00.

Kapur, S. and Remington, G. (1996). Serotonin-dopamine interaction and its relevance to schizophrenia. *Am J Psychiatry*, 153(4):466–76.

Kendler, K. S., Karkowski, L. M., and Prescott, C. A. (1999). Causal relationship between stressful life events and the onset of major depression. *Am. J. Psychiatry*, 156:837–41.

Maier, S. and Seligman, M. (1976). Learned Helplessness: Theory and Evidence. *Journal of Experimental Psychology: General*, 105(1):3–46.

Maier, S. F. and Watkins, L. R. (2005). Stressor controllability and learned helplessness: the roles of the dorsal raphe nucleus, serotonin, and corticotropin-releasing factor. *Neurosci. Biobehav. Rev.*, 29(4-5):829–41.

Montague, P. R., Dayan, P., and Sejnowski, T. J. (1996). A framework for mesencephalic dopamine systems based on predictive hebbian learning. *J. Neurosci.*, 16(5):1936–47.

Moutoussis, M., Bentall, R. P., Williams, J., and Dayan, P. (2008). A temporal difference account of avoidance learning. *Network*, 19(2):137–160.

Peterson, C., Maier, S. F., and Seligman, M. E. P. (1993). *Learned Helplessness: A theory for the age of personal control*. OUP, Oxford, UK.

Pizzagalli, D. A., Jahn, A. L., and O'Shea, J. P. (2005). Toward an objective characterization of an anhedonic phenotype: a signal-detection approach. *Biol Psychiatry*, 57(4):319–327.

Power, M., editor (2005). *Mood Disorders: A Handbook of Science and Practice*. John Wiley and Sons, paperback edition.

Smith, A., Li, M., Becker, S., and Kapur, S. (2004). A model of antipsychotic action in conditioned avoidance: a computational approach. *Neuropsychopharm.*, 29(6):1040–9.

Smith, K. A., Morris, J. S., Friston, K. J., Cowen, P. J., and Dolan, R. J. (1999). Brain mechanisms associated with depressive relapse and associated cognitive impairment following acute tryptophan depletion. *Br. J. Psychiatry*, 174:525–9.

Veiel, H. O. F. (1997). A preliminary profile of neuropsychological deficits associated with major depression. *J. Clin. Exp. Neuropsychol.*, 19:587–603.

Williams, J. and Dayan, P. (2005). Dopamine, learning, and impulsivity: a biological account of attention-deficit/hyperactivity disorder. *J Child Adolesc Psychopharmacol*, 15(2):160–79; discussion 157–9.

Williams, J. M. G. (1992). *The psychological treatment of depression*. Routledge.

Willner, P. (1997). Validity, reliability and utility of the chronic mild stress model of depression: a 10-year review and evaluation. *Psychopharm*, 134:319–29.

Yu, A. J. and Dayan, P. (2005). Uncertainty, neuromodulation, and attention. *Neuron*, 46(4):681–692.

